# Learning to combine foveal glimpses with a third-order Boltzmann machine

**Hugo Larochelle and Geoffrey Hinton**
Department of Computer Science, University of Toronto
6 King's College Rd, Toronto, ON, Canada, M5S 3G4
{larocheh,hinton}@cs.toronto.edu

## Abstract

We describe a model based on a Boltzmann machine with third-order connections that can learn how to accumulate information about a shape over several fixations. The model uses a retina that only has enough high resolution pixels to cover a small area of the image, so it must decide on a sequence of fixations and it must combine the "glimpse" at each fixation with the location of the fixation before integrating the information with information from other glimpses of the same object. We evaluate this model on a synthetic dataset and two image classification datasets, showing that it can perform at least as well as a model trained on whole images.

## 1 Introduction

Like insects with unmovable compound eyes, most current computer vision systems use images of uniform resolution. Human vision, by contrast, uses a retina in which the resolution falls off rapidly with eccentricity and it relies on intelligent, top-down strategies for sequentially fixating parts of the optic array that are relevant for the task at hand. This "fixation point strategy" has many advantages:

- It allows the human visual system to achieve invariance to large scale translations by simply translating all the fixation points.
- It allows a reduction in the number of "pixels" that must be processed in parallel yet preserves the ability to see very fine details when necessary. This reduction allows the visual system to apply highly parallel processing to the sensory input produced by each fixation.
- It removes most of the force from the main argument against generative models of perception, which is that they waste time computing detailed explanations for parts of the image that are irrelevant to the task at hand. If task-specific considerations are used to select fixation points for a variable resolution retina, most of the irrelevant parts of the optic array will only ever be represented in a small number of large pixels.

If a system with billions of neurons at its disposal has adopted this strategy, the use of a variable resolution retina and a sequence of intelligently selected fixation points is likely to be even more advantageous for simulated visual systems that have to make do with a few thousand "neurons". In this paper we explore the computational issues that arise when the fixation point strategy is incorporated in a Boltzmann machine and demonstrate a small system that can make good use of a variable resolution retina containing very few pixels. There are two main computational issues:

- **What-where combination**: How can eye positions be combined with the features extracted from the retinal input (glimpses) to allow evidence for a shape to be accumulated across a sequence of fixations?
- **Where to look next**: Given the results of the current and previous fixations, where should the system look next to optimize its object recognition performance?

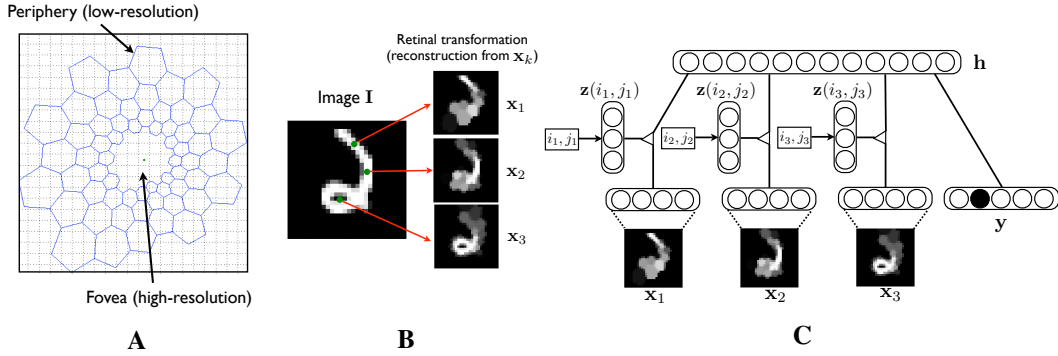

Figure 1: **A:** Illustration of the retinal transformation $\mathbf{r}(\mathbf{I}, (i, j))$. The center dot marks the pixel at position $(i, j)$ (pixels are drawn as dotted squares). **B:** examples of glimpses computed by the retinal transformation, at different positions (visualized through reconstructions). **C:** Illustration of the multi-fixation RBM.

To tackle these issues, we rely on a special type of restricted Boltzmann machine (RBM) with third-order connections between visible units (the glimpses), hidden units (the accumulated features) and position-dependent units which gate the connections between the visible and hidden units. We describe approaches for training this model to jointly learn and accumulate useful features from the image and control where these features should be extracted, and evaluate it on a synthetic dataset and two image classification datasets.

## 2 Vision as a sequential process with retinal fixations

Throughout this work, we will assume the following problem framework. We are given a training set of image and label pairs $\{(\mathbf{I}^t, l^t)\}_{t=1}^N$ and the task is to predict the value of $l^t$ (e.g. a class label $l^t \in \{1, \ldots, C\}$) given the associated image $\mathbf{I}^t$. The standard machine learning approach would consist in extracting features from the whole image $\mathbf{I}^t$ and from those directly learn to predict $l^t$. However, since we wish to incorporate the notion of fixation into our problem framework, we need to introduce some constraints on how information from $\mathbf{I}^t$ is acquired.

To achieve this, we require that information about an image $\mathbf{I}$ (removing the superscript $t$ for simplicity) must be acquired sequentially by fixating (or querying) the image at a series of $K$ positions $[(i_1, j_1), \ldots, (i_K, j_K)]$. Given a position $(i_k, j_k)$, which identifies a pixel $\mathrm{I}(i_k, j_k)$ in the image, information in the neighborhood of that pixel is extracted through what we refer to as a **retinal transformation** $\mathbf{r}(\mathbf{I}, (i_k, j_k))$. Much like the fovea of the human retina, this transformation extracts high-resolution information (i.e. copies the value of the pixels) from the image only in the neighborhood of pixel $\mathrm{I}(i_k, j_k)$. At the periphery of the retina, lower-resolution information is extracted by averaging the values of pixels falling in small hexagonal regions of the image. The hexagons are arranged into a spiral, with the size of the hexagons increasing with the distance from the center $(i_k, j_k)$ of the fixation[1]. All of the high-resolution and low-resolution information is then concatenated into a single vector given as output by $\mathbf{r}(\mathbf{I}, (i_k, j_k))$. An illustration of this retinal transformation is given in Figure 1. As a shorthand, we will use $\mathbf{x}_k$ to refer to the glimpse given by the output of the retinal transformation $\mathbf{r}(\mathbf{I}, (i_k, j_k))$.

## 3 A multi-fixation model

We now describe a system that can predict $l$ from a few glimpses $\mathbf{x}_1, \ldots, \mathbf{x}_K$. We know that this problem is solvable: [1] demonstrated that people can "see" a shape by combining information from multiple glimpses through a hole that is much smaller than the whole shape. He called this "anorthoscopic perception". The shape information derived from each glimpse cannot just be added

as implied in [2]. It is the *conjunction* of the shape of a part and its relative location that provides evidence for the shape of a whole object, and the natural way to deal with this conjunction is to use multiplicative interactions between the "what" and "where".

Learning modules that incorporate multiplicative interactions have recently been developed [3, 4]. These can be viewed as energy-based models with three-way interactions. In this work, we build on [5, 6] who introduced a method of keeping the number of parameters under control when incorporating such high-order interactions in a restricted Boltzmann machine. We start by describing the standard RBM model for classification, and then describe how we adapt it to the multi-fixation framework.

## 3.1 Restricted Boltzmann Machine for classification

RBMs are undirected generative models which model the distribution of a visible vector $\mathbf{v}$ of units using a hidden vector of binary units $\mathbf{h}$. For a classification problem with $C$ classes, the visible layer is composed of an input vector $\mathbf{x}$ and a target vector $\mathbf{y}$, where the target vector follows the so-called "1 out of $C$" representation of the classification label $l$ (i.e. $\mathbf{y} = \mathbf{e}_l$ where all the components of $\mathbf{e}_l$ are 0 except for the $l^{\text{th}}$ which is 1).

More specifically, given the following energy function:
$$E(\mathbf{y}, \mathbf{x}, \mathbf{h}) \quad = \quad -\mathbf{h}^\top \mathbf{W} \mathbf{x} - \mathbf{b}^\top \mathbf{x} - \mathbf{c}^\top \mathbf{h} - \mathbf{d}^\top \mathbf{y} - \mathbf{h}^\top \mathbf{U} \mathbf{y} \tag{1}$$
we define the associated distribution over $\mathbf{x}$, $\mathbf{y}$ and $\mathbf{h}$: $p(\mathbf{y}, \mathbf{x}, \mathbf{h}) = \exp(-E(\mathbf{y}, \mathbf{x}, \mathbf{h}))/Z$.

Assuming $\mathbf{x}$ is a binary vector, it can be shown that this model has the following posteriors:
$$p(\mathbf{h}|\mathbf{y}, \mathbf{x}) \quad = \quad \prod_j p(h_j|\mathbf{y}, \mathbf{x}), \quad \text{where} \quad p(h_j = 1|\mathbf{y}, \mathbf{x}) = \text{sigm}(c_j + \mathbf{U}_{j\cdot}\mathbf{y} + \mathbf{W}_{j\cdot}\mathbf{x}) \tag{2}$$

$$p(\mathbf{x}|\mathbf{h}) \quad = \quad \prod_i p(x_i|\mathbf{h}), \quad \text{where} \quad p(x_i = 1|\mathbf{h}) = \text{sigm}(b_i + \mathbf{h}^\top \mathbf{W}_{\cdot i}) \tag{3}$$

$$p(\mathbf{y} = \mathbf{e}_l|\mathbf{h}) \quad = \quad \frac{\exp(d_l + \mathbf{h}^\top \mathbf{U}_{\cdot l})}{\sum_{l^*=1}^C \exp(d_{l^*} + \mathbf{h}^\top \mathbf{U}_{\cdot l^*})} \tag{4}$$

where $\mathbf{A}_{j\cdot}$ and $\mathbf{A}_{\cdot i}$ respectively refer to the $j^{\text{th}}$ row and $i^{\text{th}}$ column of matrix $\mathbf{A}$. These posteriors make it easy to do inference or sample from the model using Gibbs sampling. For real-valued input vectors, an extension of Equation 1 can be derived to obtain a Gaussian distribution for the conditional distribution over $\mathbf{x}$ of Equation 3 [7].

Another useful property of this model is that all hidden units can be marginalized over analytically in order to exactly compute
$$p(\mathbf{y} = \mathbf{e}_l|\mathbf{x}) \quad = \quad \frac{\exp(d_l + \sum_j \text{softplus}(c_j + U_{jl} + \mathbf{W}_{j\cdot}\mathbf{x}))}{\sum_{l^*=1}^C \exp(d_{l^*} + \sum_j \text{softplus}(c_j + U_{jl^*} + \mathbf{W}_{j\cdot}\mathbf{x}))} \tag{5}$$
where $\text{softplus}(a) = \log(1 + \exp(a))$. Hence, classification can be performed for some given input $\mathbf{x}$ by computing Equation 5 and choosing the most likely class.

## 3.2 Multi-fixation RBM

At first glance, a very simple way of using the classification RBM of the previous section in the multi-fixation setting would be to set $\mathbf{x} = \mathbf{x}_{1:K} = [\mathbf{x}_1, \ldots, \mathbf{x}_K]$. However, doing so would completely throw away the information about the position of the fixations. Instead, we could redefine the energy function of Equation 1 as follows:
$$E(\mathbf{y}, \mathbf{x}_{1:K}, \mathbf{h}) \quad = \quad \left( \sum_{k=1}^K -\mathbf{h}^\top \mathbf{W}^{(i_k, j_k)} \mathbf{x}_k - \mathbf{b}^\top \mathbf{x}_k \right) - \mathbf{c}^\top \mathbf{h} - \mathbf{d}^\top \mathbf{y} - \mathbf{h}^\top \mathbf{U} \mathbf{y} \tag{6}$$
where the connection matrix $\mathbf{W}^{(i_k, j_k)}$ now depends on the position of the fixation[2]. Such connections are called high-order (here third order) because they can be seen as connecting the hidden

units, input units and implicit position units (one for each possible value of positions $(i_k, j_k)$). Conditioned on the position units (which are assumed to be given), this model is still an RBM satisfying the traditional conditional independence properties between the hidden and visible units.

For a given $m \times m$ grid of possible fixation positions, all $\mathbf{W}^{(i_k, j_k)}$ matrices contain $m^2 H R$ parameters where $H$ is the number of hidden units and $R$ is the size of the retinal transformation. To reduce that number, we parametrize or factorize the $\mathbf{W}^{(i_k, j_k)}$ matrices as follows

$$\mathbf{W}^{(i_k, j_k)} = \mathbf{P} \operatorname{diag}(\mathbf{z}(i_k, j_k)) \mathbf{F} \tag{7}$$

where $\mathbf{F}$ is $R \times D$, $\mathbf{P}$ is $D \times H$, $\mathbf{z}(i_k, j_k)$ is a (learned) vector associated to position $(i_k, j_k)$ and $\operatorname{diag}(\mathbf{a})$ is a matrix whose diagonal is the vector $\mathbf{a}$. Hence, $\mathbf{W}^{(i_k, j_k)}$ is now an outer product of the $D$ lower-dimensional bases in $\mathbf{F}$ ("filters") and $\mathbf{P}$ ("pooling"), gated by a position specific vector $\mathbf{z}(i_k, j_k)$. Instead of learning a separate matrix $\mathbf{W}^{(i_k, j_k)}$ for each possible position, we now only need to learn a separate vector $\mathbf{z}(i_k, j_k)$ for each position. Intuitively, the vector $\mathbf{z}(i_k, j_k)$ controls which rows of $\mathbf{F}$ and columns of $\mathbf{P}$ are used to accumulate the glimpse at position $(i_k, j_k)$ into the hidden layer of the RBM. A similar factorization has been used by [8]. We emphasize that $\mathbf{z}(i_k, j_k)$ is not stochastic but is a deterministic function of position $(i_k, j_k)$, trained by backpropagation of gradients from the multi-fixation RBM learning cost. In practice, we force the components of $\mathbf{z}(i_k, j_k)$ to be in $[0, 1]^3$. The multi-fixation RBM is illustrated in Figure 1.

## 4 Learning in the multi-fixation RBM

The multi-fixation RBM must learn to accumulate useful features from each glimpse, and it must also learn a good policy for choosing the fixation points. We refer to these two goals as "learning the what-where combination" and "learning where to look".

### 4.1 Learning the what-where combination

For now, let's assume that we are given the sequence of glimpses $\mathbf{x}_{1:K}^t$ fed to the multi-fixation RBM for each image $\mathbf{I}^t$. As suggested by [9], we can train the RBM to minimize the following hybrid cost over each input $\mathbf{x}_{1:K}^t$ and label $l^t$:

$$\textbf{Hybrid cost:} \qquad \mathcal{C}_{\text{hybrid}} = -\log p(\mathbf{y}^t | \mathbf{x}_{1:K}^t) - \alpha \log p(\mathbf{y}^t, \mathbf{x}_{1:K}^t) \tag{8}$$

where $\mathbf{y}^t = \mathbf{e}_{l^t}$. The first term in $\mathcal{C}_{\text{hybrid}}$ is the discriminative cost and its gradient with respect to the RBM parameters can be computed exactly, since $p(\mathbf{y}^t | \mathbf{x}_{1:K}^t)$ can be computed exactly (see [9] for more details on how to derive these gradients). The second term is the generative cost and its gradient can only be approximated. Contrastive Divergence [10] based on one full step of Gibbs sampling provides a good enough approximation. The RBM is then trained by doing stochastic or mini-batch gradient descent on the hybrid cost.

In [9], it was observed that there is typically a value of $\alpha$ which yields better performance than using either discriminative or generative costs alone. Putting more emphasis on the discriminative term ensures that more capacity is allocated to predicting the label values than to predicting each pixel value, which is important because there are many more pixels than labels. The generative term acts as a data-dependent regularizer that encourages the RBM to extract features that capture the statistical structure of the input. This is a much better regularizer than the domain-independent priors implemented by L1 or L2 regularization.

We can also take advantage of the following obvious fact: If the sequence $\mathbf{x}_{1:K}^t$ is associated with a particular target label $\mathbf{y}^t$, then so are all the subsequences $\mathbf{x}_{1:k}^t$ where $k < K$. Hence, we can also train the multi-fixation RBM on these subsequences using the following "hybrid-sequential" cost:

$$\textbf{Hybrid-sequential cost:} \; \mathcal{C}_{\text{hybrid-seq}} = \sum_{k=1}^{K} -\log p(\mathbf{y}^t | \mathbf{x}_{1:k}^t) - \alpha \log p(\mathbf{y}^t, \mathbf{x}_k^t | \mathbf{x}_{1:k-1}^t) \tag{9}$$

where the second term, which corresponds to negative log-likelihoods under a so-called conditional RBM [8], plays a similar role to the generative cost term of the hybrid cost and encourages the

RBM to learn about the statistical structure of the input glimpses. An estimate of the gradient of this term can also be obtained using Contrastive Divergence (see [8] for more details). While being more expensive than the hybrid cost, the hybrid-sequential cost could yield better generalization performance by better exploiting the training data. Both costs are evaluated in Section 6.1.

## 4.2 Learning where to look

Now that we have a model for processing the glimpses resulting from fixating at different positions, we need to define a model which will determine where those fixations should be made on the $m \times m$ grid of possible positions.

After $k - 1$ fixations, this model should take as input some vector $\mathbf{s}_k$ containing information about the glimpses accumulated so far (e.g. the current activation probabilities of the multi-fixation RBM hidden layer), and output a score $f(\mathbf{s}_k, (i_k, j_k))$ for each possible fixation position $(i_k, j_k)$. This score should be predictive of how useful fixating at the given position will be. We refer to this model as the **controller**.

Ideally, the fixation position with highest score under the controller should be the one which maximizes the chance of correctly classifying the input image. For instance, a good controller could be such that

$$f(\mathbf{s}_k, (i_k, j_k)) \quad \propto \quad \log p(\mathbf{y}^t | \mathbf{x}_{1:k-1}^t, \mathbf{x}_k^t = \mathbf{r}(\mathbf{I}, (i_k, j_k))) \tag{10}$$

i.e. its output is proportional to the log-probability the RBM will assign to the **true** target $\mathbf{y}^t$ of the image $\mathbf{I}^t$ once it has fixated at position $(i_k, j_k)$ and incorporated the information in that glimpse. In other words, we would like the controller to assign high scores to fixation positions which are more likely to provide the RBM with the necessary information to make a correct prediction of $\mathbf{y}^t$.

A simple training cost for the controller could then be to reduce the absolute difference between its prediction $f(\mathbf{s}_k, (i_k, j_k))$ and the observed value of $\log p(\mathbf{y}^t | \mathbf{x}_{1:k-1}^t, \mathbf{x}_k = \mathbf{r}(\mathbf{I}, (i_k, j_k)))$ for the sequences of glimpses generated while training the multi-fixation RBM. During training, these sequences of glimpses can be generated from the controller using the Boltzmann distribution

$$p_{\text{controller}}((i_k, j_k) | \mathbf{x}_{1:k-1}^t) \quad \propto \quad \exp(f(\mathbf{s}_k, (i_k, j_k))) \tag{11}$$

which ensures that all fixation positions can be sampled but those which are currently considered more useful by the controller are also more likely to be chosen. At test time however, for each $k$, the position that is the most likely under the controller is chosen[4].

In our experiments, we used a linear model for $f(\mathbf{s}_k, (i_k, j_k))$, with separate weights for each possible value of $(i_k, j_k)$. The controller is the same for all $k$, i.e. $f(\mathbf{s}_k, (i_k, j_k))$ only depends on the values of $\mathbf{s}_k$ and $(i_k, j_k)$ (though one could consider training a separate controller for each $k$). A constant learning rate of 0.001 was used for training. As for the value taken by $\mathbf{s}_k$, we set it to

$$\text{sigm}\left(\mathbf{c} + \sum_{k^*=1}^{k-1} \mathbf{W}^{(i_{k^*}, j_{k^*})} \mathbf{x}_{k^*}\right) \quad = \quad \text{sigm}\left(\mathbf{c} + \sum_{k^*=1}^{k-1} \mathbf{P} \, \text{diag}(\mathbf{z}(i_{k^*}, j_{k^*})) \, \mathbf{F} \, \mathbf{x}_{k^*}\right) \tag{12}$$

which can be seen as an estimate of the probability vector for each hidden unit of the RBM to be 1, given the previous glimpses $\mathbf{x}_{1:k-1}$. For the special case $k = 1$, $\mathbf{s}_1$ is computed based on a fixation at the center of the image but all the information in this initial glimpse is then "forgotten", i.e. it is only used for choosing the first image-dependent fixation point and is not used by the multi-fixation RBM to accumulate information about the image. We also concatenate to $\mathbf{s}_k$ a binary vector of size $m^2$ (one component for each possible fixation position), where a component is 1 if the associated position has been fixated. Finally, in order to ensure that a fixation position is never sampled twice, we impose that $p_{\text{controller}}((i_k, j_k) | \mathbf{x}_{1:k-1}^t) = 0$ for all positions previously sampled.

## 4.3 Putting it all together

Figure 2 summarizes how the multi-fixation RBM and the controller are jointly trained, for either the hybrid cost or the hybrid-sequential cost. Details on gradient computations for both costs are

also given in the supplementary material. To our knowledge, this is the first implemented system for combining glimpses that jointly trains a recognition component (the RBM) with an attentional component (the fixation controller).

## 5    Related work

A vast array of work has been dedicated to modelling the visual search behavior of humans [11, 12, 13, 14], typically through the computation of saliency maps [15, 16]. Most of such work, however, is concerned with the prediction of salient regions in an image, and not with the other parts of a task-oriented vision classifier.

Surprisingly little work has been done on how best to combine multiple glimpses in a recognition system. SIFT features have been proposed either as a prefilter for reducing the number of possible fixation positions [17] or as a way of preprocessing the raw glimpses [13]. [18] used a fixed and hand-tuned saliency map to sample small patches in images of hand-written characters and trained a recursive neural network from sequences of such patches. By contrast, the model proposed here does not rely on hand-tuned features or saliency maps and learns from scratch both the where to look and what-where combination components. A further improvement on the aforecited work consists in separately learning both the where to look and the what-where combination components [19, 20]. In this work however, both components are learned jointly, as opposed to being put together only at test time. For instance, [19] use a saliency map based on filters previously trained on natural images for the where to look component, and the what-where combination component for recognition is a nearest neighbor density estimator. Moreover, their goal is not to avoid fixating everywhere, but to obtain more robust recognition by using a saliency map (whose computation effectively corresponds to fixating everywhere in the image). In that respect, our work is orthogonal, as we are treating each fixation as a costly operation (e.g. we considered up to 6 fixations, while they used 100 fixations).

## 6    Experiments

We present three experiments on three different image classification problems. The first is based on the MNIST dataset and is meant to evaluate the multi-fixation RBM alone (i.e. without the controller). The second is on a synthetic dataset and is meant to analyze the controller learning algorithm and its interaction with the multi-fixation RBM. Finally, results on a facial expression recognition problem are presented.

### 6.1    Experiment 1: Evaluation of the multi-fixation RBM

In order to evaluate the multi-fixation RBM of Section 3.2 separately from the controller model, we trained a multi-fixation RBM[5] on a fixed set of 4 fixations (i.e. the same fixation positions for all images). Those fixations were centered around the pixels at positions $\{(9, 9), (9, 19), (19, 9), (19, 19)\}$ (MNIST images are of size $28 \times 28$) and their order was chosen at random for every parameter update of the RBM. The retinal transformation had a high-resolution fovea covering 38 pixels and 60 hexagonal low-resolution regions in the periphery (see Figure 2 for an illustration). We used the training, validation and test splits proposed by [21], with a training set of 10 000 examples.

The results are given in Figure 2, with comparisons with an RBF kernel SVM classifier and a single hidden layer neural network initialized using unsupervised training of an RBM on the training set (those two baselines were trained on the full MNIST images). The multi-fixation RBM yields performance comparable to the baselines despite only having four glimpses, and the hybrid-sequential cost function works better than the non-sequential, hybrid cost.

### 6.2    Experiment 2: evaluation of the controller

In this second experiment, we designed a synthetic problem where the optimal fixation policy is known, to validate the proposed training algorithm for the controller. The task is to identify whether

**Pseudocode for training update**
· compute $\mathbf{s}_1$ based on center of image
**for** $k$ from 1 to $K$ **do**
  · sample $(i_k, j_k)$ from $p_{\text{controller}}((i_k, j_k)|\mathbf{x}_{1:k-1}^t)$
  · compute $\mathbf{x}_k = \mathbf{r}(\mathbf{I}, (i_k, j_k))$
  · update controller with a gradient step for error
    $|f(\mathbf{s}_k, (i_k, j_k)) - \log p(\mathbf{y}|\mathbf{x}_{1:k})|$
  **if** using hybrid-sequential cost **then**
    · accumulate gradient on RBM parameters
      of $k^{\text{th}}$ term in cost $\mathcal{C}_{\text{hybrid-seq}}$
  **end if**
  · compute $\mathbf{s}_{k+1}$
**end for**
**if** using hybrid-sequential cost **then**
  · update RBM parameters based on accumulated
    gradient of hybrid-sequential cost $\mathcal{C}_{\text{hybrid-seq}}$
**else** {using hybrid cost}
  · update RBM based on gradient of hybrid cost $\mathcal{C}_{\text{hybrid}}$
**end if**

**A**

### Experiment 1: MNIST with 4 fixations

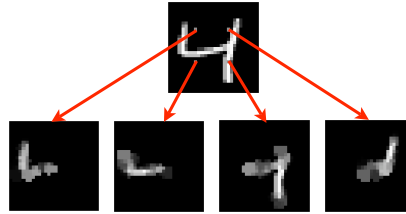

| Model | Error |
|---|---|
| NNet+RBM [22] | 3.17% ($\pm$ 0.15) |
| SVM [21] | 3.03% ($\pm$ 0.15) |
| Multi-fixation RBM (hybrid) | 3.20% ($\pm$ 0.15) |
| Multi-fixation RBM (hybrid-sequential) | 2.76% ($\pm$ 0.14) |

**B**

Figure 2: **A:** Pseudocode for the training update of the multi-fixation RBM, using either the hybrid or hybrid-sequential cost. **B:** illustration of glimpses and results for experiment on MNIST.

there is a horizontal (positive class) or vertical (negative class) 3-pixel white bar somewhere near the edge of a $15 \times 15$ pixel image. At the center of the image is one of 8 visual symbols, indicating the location of the bar. This symbol conveys no information about the class (the positive and negative classes are equiprobable) but is necessary to identify where to fixate. Figure 3 shows positive and negative examples. There are only 48 possible images and the model is trained on all of them (i.e. we are measuring the capacity of the model to learn this problem perfectly). Since, as described earlier, the input $\mathbf{s}_1$ of the controller contains information about the center of the image, only one fixation decision by the controller suffices to solve this problem.

A multi-fixation RBM was trained jointly with a controller on this problem[6], with only $K = 1$ fixation. When trained according to the hybrid cost of Equation 8 ($\alpha = 1$), the model was able to solve this problem perfectly without errors, i.e. the controller always proposes to fixate at the region containing the white bar and the multi-fixation RBM always correctly recognizes the orientation of the bar. However, using only the discriminative cost ($\alpha = 0$), it is never able to solve it (i.e. has an error rate of 50%), even if trained twice as long as for $\alpha = 1$. This is because the purely discriminative RBM never learns meaningful features for the non-discriminative visual symbol at the center, which are essential for the controller to be able to predict the position of the white bar.

### 6.3 Experiment 3: facial expression recognition experiment

Finally, we applied the multi-fixation RBM with its controller to a problem of facial expression recognition. The dataset [23] consists in 4178 images of size $100 \times 100$, depicting people acting one of seven facial expressions (anger, disgust, fear, happiness, sadness, surprise and neutral, see Figure 3 for examples). Five training, validation and test set splits where generated, ensuring that all images of a given person can only be found in one of the three sets. Pixel values of the images were scaled to the $[-0.5, 0.5]$ interval.

A multi-fixation RBM learned jointly with a controller was trained on this problem[7], with $K = 6$ fixations. Possible fixation positions were layed out every 10 pixels on a $7 \times 7$ grid, with the top-left

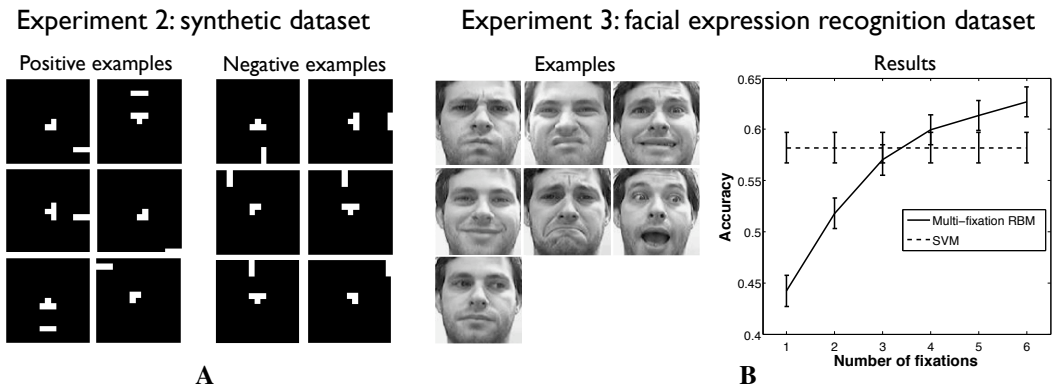

Figure 3: **A:** positive and negative from the synthetic dataset of experiment 2. **B:** examples and results for the facial expression recognition dataset.

position being at pixel $(20, 20)$. The retinal transformation covered around 2000 pixels and didn't use a periphery[8] (all pixels were from the fovea). Moreover, glimpses were passed through a "pre-processing" hidden layer of size 250, initialized by unsupervised training of an RBM with Gaussian visible units (but without target units) on glimpses from the $7 \times 7$ grid. During training of the multi-fixation RBM, the discriminative part of its gradient was also passed through the preprocessing hidden layer for fine-tuning of its parameters.

Results are reported in Figure 3, where the multi-fixation RBM is compared to an RBF kernel SVM trained on the full images. The accuracy of the RBM is given after a varying number of fixations. We can see that after 3 fixations (i.e. around 60% of the image) the multi-fixation RBM reaches a performance that is statistically equivalent to that of the SVM ($58.2 \pm 1.5\%$) trained on the full images. Training the SVM on a scaled-down version of the data ($48 \times 48$ pixels) gives a similar performance of $57.8\%$ ($\pm 1.5\%$). At 5 fixations, the multi-fixation RBM now improves on the SVM, and gets even better at 6 fixations, with an accuracy of $62.7\%$ ($\pm 1.5\%$). Finally, we also computed the performance of a linear SVM classifier trained on the concatenation of the hidden units from a unique RBM with Gaussian visible units applied at all $7 \times 7$ positions (the same RBM used for initializing the preprocessing layer of the multi-fixation RBM was used). This convolutional approach, which requires 49 fixations, yields a performance of $61.2\%$ ($\pm 1.5\%$), slightly worse but statistically indistinguishable from the multi-fixation RBM which only required 6 fixations.

## 7   Conclusion

Human vision is a sequential sampling process in which only a fraction of the optic array is ever processed at the highest resolution. Most computer vision work on object recognition ignores this fact and can be viewed as modelling tachistoscopic recognition of very small objects that lie entirely within the fovea. We have focused on the other extreme, i.e. recognizing objects by using multiple task-specific fixations of a retina with few pixels, and obtained positive results. We believe that the intelligent choice of fixation points and the integration of multiple glimpses will be essential for making biologically inspired vision systems work well on large images.

**Acknowledgments**
We thank Marc'Aurelio Ranzato and the reviewers for many helpful comments, and Josh Susskind and Tommy Liu for help with the facial expression dataset. This research was supported by NSERC.

## Footnotes

[1]A retina with approximately hexagonal pixels produced by a log conformal mapping centered on the current fixation point has an interesting property: It is possible to use weight-sharing over scale and orientation instead of translation, but we do not explore this here.

[2]To be strictly correct in our notation, we should add the position coordinates $(i_1, j_1), \ldots, (i_K, j_K)$ as an input of the energy function $E(\mathbf{y}, \mathbf{x}, \mathbf{h})$. To avoid clutter however, we will consider the position coordinates to be implicitly given by $\mathbf{x}_1, \ldots, \mathbf{x}_K$.

[3]This is done by setting $\mathbf{z}(i_k, j_k) = \operatorname{sigm}(\bar{\mathbf{z}}(i_k, j_k))$ and learning the unconstrained $\bar{\mathbf{z}}(i_k, j_k)$ vectors instead. We also use a learning rate 100 times larger for learning those parameters.

[4]While it might not be optimal, this greedy search for the best sequence of fixation positions is simple and worked well in practice.

[5]The RBM used $H = 500$ hidden units and was trained with a constant learning rate of 0.1 (no momentum was used). The learned position vectors $\mathbf{z}(i_k, j_k)$ were of size $D = 250$. Training lasted for 2000 iterations, with a validation set used to keep track of generalization performance and remember the best parameter value of the RBM. We report results when using either the hybrid cost of Equation 8 or the hybrid-sequential cost of Equation 9, with $\alpha = 0.1$. Mini-batches of size 100 were used.

[6]Hyper-parameters: $H = 500$, $D = 250$. Stochastic gradient descent was used with a learning rate of 0.001. The controller had the choice of 9 possible fixation positions, each covering either one of the eight regions where bars can be found or the middle region where the visual symbol is. The retinal transformation was such that information from only one of those regions is transferred.

[7]Hyper-parameters: $H = 250$, $D = 250$. Stochastic gradient descent was used with a learning rate of 0.01. The RBM was trained with the hybrid cost of Equation 8 with $\alpha = 0.001$ (the hybrid cost was preferred mainly because it is faster). Also, the matrix $\mathbf{P}$ was set to the identity matrix and only $\mathbf{F}$ was learned (this removed a matrix multiplication and thus accelerated learning in the model, while still giving good results). The vectors

## References

[1]  Hermann von Helmholtz. *Treatise on physiological optics*. Dover Publications, New York, 1962.

---

$\mathbf{z}(i, j)$ were initialized in a topographic manner (i.e. each component of $\mathbf{z}(i, j)$ is $\gg 0$ only in a small region of the image). Finally, to avoid overfitting, exponentially decaying averages of the parameters of the model were maintained throughout training and were used as the values of the model at test time.

[8]This simplification of the retinal transformation makes it more convenient to estimate the percentage of high-resolution pixels used by the multi-fixation RBM and contrast it with the SVM trained on the full image.

[2] Arash Fazl, Stephen Grossberg, and Ennio Mingolla. View-invariant object category learning, recognition, and search: how spatial and object attention are coordinated using surface-based attentional shrouds. *Cogn Psychol*, 58(1):1–48, 2009.

[3] Roland Memisevic and Geoffrey E. Hinton. Unsupervised learning of image transformations. In *In Computer Vision and Pattern Recognition. IEEE Computer Society*, 2007.

[4] Urs Köster and Aapo Hyvärinen. A two-layer ica-like model estimated by score matching. In *ICANN'07: Proceedings of the 17th international conference on Artificial neural networks*, pages 798–807, Berlin, Heidelberg, 2007. Springer-Verlag.

[5] Geoffrey E. Hinton. Learning to represent visual input. *Phil. Trans. R. Soc.*, 365(1537):177–84, 2010.

[6] Roland Memisevic and Geoffrey E. Hinton. Learning to represent spatial transformations with factored higher-order boltzmann machines. *Neural Computation*, 22:1473–1492, 2010.

[7] Geoffrey E. Hinton and Ruslan Salakhutdinov. Reducing the dimensionality of data with neural networks. *Science*, 313(5786):504–507, July 2006.

[8] Graham W. Taylor and Geoffrey E. Hinton. Factored conditional restricted boltzmann machines for modeling motion style. In *ICML '09: Proceedings of the 26th Annual International Conference on Machine Learning*, pages 1025–1032, New York, NY, USA, 2009. ACM.

[9] Hugo Larochelle and Yoshua Bengio. Classification using discriminative restricted boltzmann machines. In *ICML '08: Proceedings of the 25th international conference on Machine learning*, pages 536–543, New York, NY, USA, 2008. ACM.

[10] Geoffrey E. Hinton. Training products of experts by minimizing contrastive divergence. *Neural Computation*, 14:1771–1800, 2002.

[11] Rajesh P.N. Rao, Gregory J. Zelinsky, Mary M. Hayhoe, and Dana H. Ballard. Modeling saccadic targeting in visual search. In David S. Touretzky, Michael Mozer, and Michael E. Hasselmo, editors, *Advances in Neural Information Processing Systems 8*, pages 830–836. MIT Press, 1996.

[12] Laura Walker Renninger, James M. Coughlan, Preeti Verghese, and Jitendra Malik. An information maximization model of eye movements. In Lawrence K. Saul, Yair Weiss, and Léon Bottou, editors, *Advances in Neural Information Processing Systems 17*, pages 1121–1128. MIT Press, Cambridge, MA, 2005.

[13] Wei Zhang, Hyejin Yang, Dimitris Samaras, and Gregory Zelinsky. A computational model of eye movements during object class detection. In Y. Weiss, B. Schölkopf, and J. Platt, editors, *Advances in Neural Information Processing Systems 18*, pages 1609–1616. MIT Press, Cambridge, MA, 2006.

[14] Antonio Torralba, Monica S. Castelhano, Aude Oliva, and John M. Henderson. Contextual guidance of eye movements and attention in real-world scenes: the role of global features in object search. *Psychological Review*, 113:2006, 2006.

[15] Laurent Itti, Christof Koch, and Ernst Niebur. A model of saliency-based visual attention for rapid scene analysis. *IEEE Trans. Pattern Anal. Mach. Intell.*, 20(11):1254–1259, 1998.

[16] Laurent Itti and Christof Koch. Computational modelling of visual attention. *Nature Reviews Neuroscience*, 2(3):194–203, 2001.

[17] Lucas Paletta, Gerald Fritz, and Christin Seifert. Q-learning of sequential attention for visual object recognition from informative local descriptors. In *ICML '05: Proceedings of the 22nd international conference on Machine learning*, pages 649–656, New York, NY, USA, 2005. ACM.

[18] Ethem Alpaydin. Selective attention for handwritten digit recognition. In David S. Touretzky, Michael Mozer, and Michael E. Hasselmo, editors, *Advances in Neural Information Processing Systems 8*, pages 771–777. MIT Press, 1996.

[19] Christopher Kanan and Garrison Cottrell. Robust classification of objects, faces, and flowers using natural image statistics. In *CVPR*, 2010.

[20] Stephen Gould, Joakim Arfvidsson, Adrian Kaehler, Benjamin Sapp, Marius Messner, Gary Bradski, Paul Baumstarck, Sukwon Chung, and Andrew Y. Ng. Peripheral-foveal vision for real-time object recognition and tracking in video. In *In International Joint Conference on Artificial Intelligence (IJCAI*, 2007.

[21] Hugo Larochelle, Dumitru Erhan, Aaron Courville, James Bergstra, and Yoshua Bengio. An empirical evaluation of deep architectures on problems with many factors of variation. In *ICML '07: Proceedings of the 24th international conference on Machine learning*, pages 473–480, New York, NY, USA, 2007. ACM.

[22] Hugo Larochelle, Yoshua Bengio, Jerome Louradour, and Pascal Lamblin. Exploring strategies for training deep neural networks. *Journal of Machine Learning Research*, 10:1–40, 2009.

[23] Josh M. Susskind, Adam K. Anderson, and Geoffrey E. Hinton. The toronto face database. Technical Report UTML TR 2010-001, Dept. of Computer Science, University of Toronto, 2010.

